# Neural Network Weight Matrix Synthesis Using Optimal Control Techniques

O. Farotimi

A. Dembo
Information Systems Lab.
Electrical Engineering Dept.
Stanford University,
Stanford, CA 94305

T. Kailath

## ABSTRACT

Given a set of input-output training samples, we describe a procedure for determining the time sequence of weights for a dynamic neural network to model an arbitrary input-output process. We formulate the input-output mapping problem as an optimal control problem, defining a performance index to be minimized as a function of time-varying weights. We solve the resulting nonlinear two-point-boundary-value problem, and this yields the training rule. For the performance index chosen, this rule turns out to be a continuous time generalization of the outer product rule earlier suggested heuristically by Hopfield for designing associative memories. Learning curves for the new technique are presented.

## 1  INTRODUCTION

Suppose that we desire to model as best as possible some unknown map $\phi : \mathcal{U} \to \mathcal{V}$, where $\mathcal{U}, \mathcal{V} \subseteq \mathcal{R}^n$. One way we might go about doing this is to collect as many input-output samples $\{(\theta_{in}, \theta_{out}) : \phi(\theta_{in}) = \theta_{out}\}$ as possible and "find" some function $f : \mathcal{U} \to \mathcal{V}$ such that a suitable distance metric $d(f(x(t)), \phi(x(t)))|_{x \in \{\theta_{in} : \phi(\theta_{in}) = \theta_{out}\}}$ is minimized.

In the foregoing, we assume a system of ordinary differential equations motivated by dynamic neural network structures[1] [2]. In particular we set up an $n$-dimensional

neural network; call it $\mathcal{N}$. Our goal is to synthesize a possibly time varying weight matrix for $\mathcal{N}$ such that for initial conditions $x(t_0)$, the input-output transformation, or flow $f : x(t_0) \rightarrow f(x(t_f))$ associated with $\mathcal{N}$ approximates closely the desired map $\phi$.

For the purposes of synthesizing the weight program for $\mathcal{N}$, we consider another system, say $\mathcal{S}$, a formal $nL$-dimensional system of differential equations comprising $L$ $n$-dimensional subsystems. With the exception that all $L$ $n$-dimensional subsystems are *constrained to have the same weight matrix*, they are otherwise identical and decoupled. We shall use this system to determine the optimal weight program given $L$ input-output samples. The resulting time program of weights is then applied to the original $n$-dimensional system $\mathcal{N}$ during normal operation. We emphasize the difference between this scheme and a simple $L$-fold replication of $\mathcal{N}$: the latter will yield a practically unwieldy $nL \times nL$ weight matrix sequence, and in fact will generally not discover the underlying map from $\mathcal{U}$ to $\mathcal{V}$, discovering instead *different* maps for each input-output sample pair. By constraining the weight matrix sequence to be an identical $n \times n$ matrix for each subsystem during this synthesis phase, our scheme in essence forces the weight sequence to capture some underlying relationship between *all* the input-output pairs. This is arguably the best estimate of the map given the information we have.

Using formal optimal control techniques[3], we set up a performance index to maximize the correlation between the system $\mathcal{S}$ output and the desired output. This optimization technique leads in general to a nonlinear two-point-boundary-value problem, and is not usually solvable analytically. For this particular performance index we are able to derive an analytical solution to the optimization problem. The optimal interconnection matrix at each time is the sum (over the index of all samples) of the outer products between each desired output $n$-vector and the corresponding subsystem output. At the end of this synthesis procedure, the weight matrix sequence represents an optimal time-varying program for the weights of the $n$-dimensional neural network $\mathcal{N}$ that will approximate $\phi : \mathcal{U} \rightarrow \mathcal{V}$.

We remark that in the ideal case, the weight matrix at the final time (i.e *one* element of the time sequence) corresponds to the symmetric matrix suggested empirically by Hopfield for associative memory applications[4]. It becomes clear that the Hopfield matrix is suboptimal for associative memory, being just one point on the optimal weight trajectory; it is optimal only in the special case where the initial conditions coincide exactly with the desired output.

In Section 2 we outline the mathematical formulation and solution of the synthesis technique, and in Section 3 we present the learning curves. The learning curves also by default yield the system performance *over the training samples*, and we compare this performance to that of the outer product rule. In Section 4 we give concluding remarks and give the directions of our future work.

Although the results here are derived for a specific case of the neuron state equation, and a specific choice of performance index, in further work we have extended the results to very general state equations and performance indices.

## 2   SYNTHESIS OF WEIGHT MATRIX TIME SEQUENCE

Suppose we have a training set consisting of $L$ pairs of $n$-dimensional vectors $(\tilde{\theta}^{(r)}{}_i, \theta^{(r)}{}_i), r = 1, 2, \ldots, L, i = 1, 2, \ldots, n$. For example, in an autoassociative system in which we desire to store $\theta^{(r)}{}_i, r = 1, 2, \ldots, L, i = 1, 2, \ldots, n$, we can choose the $\tilde{\theta}^{(r)}{}_i, r = 1, 2, \ldots, L, i = 1, 2, \ldots, n$ to be sample points in the neighborhood of $\theta^{(r)}{}_i$ in $n$-dimensional space. The idea here is that by training the network to map samples in the neighborhood of an exemplar to the exemplar, it will have developed a map that can smoothly interpolate (or *generalize*) to other points around the exemplar that may not be in the training set. In this paper we deal with the issue of finding the weight matrix that transforms the neural network dynamics into such a map. We demonstrate through simulation results that such a map can be achieved. For autoassociation, and using error vectors drawn from the training set, we show that the method here performs better (in an error-correcting sense) than the outer product rule. We are still investigating the performance of the network in generalizing to samples outside the training set.

We construct an $n$-dimensional *neural network* system $\mathcal{N}$ to model the underlying input-output map according to

$$\mathcal{N}: \quad \dot{x}(t) = -x(t) + W(t)g(x(t)), \tag{1}$$

We interpret $x$ as the neuron activation, $g(x(t))$ is the neuron output, and $W(t)$ is the neural network weight matrix.

To determine the appropriate $W(t)$, we define an $nL$-dimensional *formal* system of differential equations, $\mathcal{S}$

$$\mathcal{S}: \quad \dot{x}^*(t) = -x_*(t) + W_*(t)g(x_*), \quad g(x_*(t_0)) = \tilde{\theta} \tag{2}$$

formed by concatenating the equations for $\mathcal{N}$ $L$ times. $W_*(t)$ is block-diagonal with *identical* blocks $W(t)$. $\theta$ is the concatenated vector of sample desired outputs, $\tilde{\theta}$ is the concatenated vector of sample inputs.

The performance index for $\mathcal{S}$ is

$$\min_{w_j} J = \min_{w_j} \left\{ -x_*{}^T(t_f)\theta + \frac{1}{2}\int_{t_0}^{t_f} \left( -2x_*{}^T(t)\theta + \beta Q + \beta^{-1}\sum_{j=1}^{n} w_j^T(t)w_j(t) \right) dt \right\} \tag{3}$$

The performance index is chosen to *minimize the negative of the correlation* between the (concatenated) neuron activation and the (concatenated) desired output vectors, or equivalently *maximize the correlation* between the activation and the desired output at the final time $t_f$, (the term $-x_*{}^T(t_f)\theta$). Along the way from initial time $t_0$ to final time $t_f$, the term $-x_*{}^T(t)\theta$ under the integral penalizes decorrelation of the neuron activation and the desired output. $w_j(t), j = 1, 2, \ldots, n$ are the rows of $W(t)$, and $\beta$ is a positive constant. The term $\beta^{-1}\sum_{j=1}^{n} w_j^T(t)w_j(t)$ effects a bound

on the magnitude of the weights. The term

$$Q(g(\boldsymbol{x}(t))) = \sum_{j=1}^{n}\sum_{r=1}^{L}\sum_{u=1}^{n}\sum_{v=1}^{L} \theta_j{}^{(r)}\theta_j{}^{(v)} g(\boldsymbol{x}_u{}^{(v)}) g(\boldsymbol{x}_u{}^{(r)}),$$

and its meaning will be clear when we examine the optimal path later. $g(\cdot)$ is assumed $C^1$ differentiable.

Proceeding formally[3], we define the *Hamiltonian*:

$$
\begin{aligned}
H &= \frac{1}{2}\left(-2\boldsymbol{x}^T(t)\theta + Q + \sum_{j=1}^{n} \boldsymbol{w}_j^T(t)\boldsymbol{B}\boldsymbol{w}_j(t)\right) + \boldsymbol{\lambda}^T(t)\left(-\boldsymbol{x}(t) + \boldsymbol{W}_*(t)g(\boldsymbol{x}(t))\right) \qquad (4) \\
&= \frac{1}{2}\left(-2\boldsymbol{x}^T(t)\theta + Q + \sum_{j=1}^{n} \boldsymbol{w}_j^T(t)\boldsymbol{B}\boldsymbol{w}_j(t)\right) - \boldsymbol{\lambda}^T(t)\boldsymbol{x}(t) + \sum_{r=1}^{L}\sum_{j=1}^{n}\lambda^{(r)}{}_j \boldsymbol{w}_j^T(t)g^{(r)}(\boldsymbol{x}(t))
\end{aligned}
$$

where

$$\boldsymbol{\lambda}^T(t) = \left[\; \lambda^{(1)}{}_1(t) \quad \lambda^{(1)}{}_2(t) \quad \cdots \quad \lambda^{(L)}{}_n(t) \;\right]$$

is the vector of Lagrange multipliers, and we have used the fact that $\boldsymbol{W}_*(t)$ is block-diagonal with identical blocks $\boldsymbol{W}(t)$ in writing the summation of the last term in the second line of equation (4). The *Euler-Lagrange equations* are then given by

$$-\dot{\boldsymbol{\lambda}} = \left(\frac{\partial H}{\partial \boldsymbol{x}}\right)^T = \frac{1}{2}\left(\frac{\partial Q}{\partial \boldsymbol{x}}\right)^T - (\theta + \boldsymbol{\lambda}(t)) + \left(\frac{\partial g}{\partial \boldsymbol{x}}\right)^T \boldsymbol{W}_*{}^T(t)\boldsymbol{\lambda}(t) \qquad (5)$$

$$\boldsymbol{\lambda}(t_f) = -\theta \qquad (6)$$

$$0 = \frac{\partial H}{\partial \boldsymbol{w}_j} = \boldsymbol{w}_j^T \boldsymbol{B} + \sum_{r=1}^{L}\lambda^{(r)}{}_j g^{(r)}{}^T(\boldsymbol{x}(t)) \qquad (7)$$

From equation (7) we have

$$w_{ij}(t) = -\beta \sum_{r=1}^{L} \lambda^{(r)}{}_i g(\boldsymbol{x}_j{}^{(r)}(t)) \qquad (8)$$

Choosing

$$\boldsymbol{\lambda}(t) = -\theta \qquad (9)$$

satisfies the final condition (6), and with some algebra we find that this choice is also consistent with equations (5) and (7). The optimal weight program is therefore

$$w_{ij}(t) = \beta \sum_{r=1}^{L} \theta^{(r)}{}_i g(\boldsymbol{x}_j{}^{(r)}(t)) \qquad (10)$$

This describes the weight paradigm to be applied to the $n$-dimensional neural network-system $\mathcal{N}$ in order to model the underlying map described by the sample

points. A similar result can be derived for the discrete-time network $x(k + 1) = W(k)g(x(k))$:

$$w_{ij}(k) = \beta \sum_{r=1}^{L} \theta^{(r)}{}_i g(x_j^{(r)}(k))$$

## 2.1   REMARKS

- *Meaning of Q.*
  On the optimal path, using equation (10), it is straightforward to show that

$$\beta Q = \beta^{-1} \sum_{j=1}^{n} w_j^T(t)w_j(t)$$

  Thus $Q$ acts like another integral constraint term on the weights.

- *The Optimal Return Function.*
  The optimal return function[3], which is the value of the performance index on the optimal path can be shown to be

$$J^o(x_*, t) = -x_*{}^T(t)\theta$$

  Thus the optimal weight matrix $W(t)$ seeks at every instant to minimize the negative correlation (or maximize the correlation) on the optimal path in the formal system $\mathcal{S}$ (and hence in the neural network $\mathcal{N}$).

- *Comparison with outer product rule.*
  It is worthwhile to compare equation (10) with the outer product rule:

$$w_{ij} = \beta \sum_{r=1}^{L} \theta^{(r)}{}_i \theta^{(r)}{}_j \tag{11}$$

  We see that the outer product rule is just one point on the weight trajectory defined by equation (10) - the point at final time $t_f$ when $g(x_j^{(r)}(t_f)) = \theta^{(r)}{}_j$.

## 3   LEARNING CURVES

In our simulation we considered 14 8-dimensional vectors as the desired outputs. The *weight synthesis* or *learning* phase is as follows: we initialized the 112-dimensional *formal synthesis system S* with a corrupted version of the vector set, and used equation (10) to find the optimal 8 × 8 weight matrix sequence for an *8-dimensional neural network N* to correctly classify any of the corrupted 14 vectors. The weight sequence is recorded. This procedure is required *only once* for any given training set. After this learning is completed, the normal operation of the neural network $\mathcal{N}$ consists in running it using the weights obtained from the synthesis phase above. The resulting network describes a continuous input-output map. At points belonging to the training set this map coincides with the underlying map we are trying to model. For points outside the training set, it performs a nonlinear interpolation

(generalization) the nature of which is determined by the training set as well as the neuron state equation. Figure 1 shows the learning procedure through time. The curves labeled *"Optimally Trained Network"* shows the behavior of two correlation measures as the training proceeds. One correlation measure used was the cosine of the angle between the desired vector ($\theta$) and the neuron activation ($x$) vector. The other correlation measure was the cosine of the angle between the desired vector ($\theta$) and the neuron output ($g(x(t))$) vector. Given our system initialization in equation (2), the correlation $g(x(t))^T\theta$ more accurately represents our objective, although the performance index (3) reflects the correlation $x^T\theta$. The reason for our performance index choice is that the weight trajectory yielded by $g(x(t))^T\theta$ leads the system to an all-zero, trivial equilibrium for a sigmoid $g(\cdot)$ (we used such a $g(\cdot)$ with saturation values at $+1$ and $-1$ in our simulations). This is not the case for the weight trajectory yielded by $x^T\theta$. Since $g(x(t))$ is monotonic with $x$, $x^T\theta$ represented an admissible alternative choice for the performance index. The results bear this out. Another possible choice is $(g(x(t)) + x)^T\theta$. This gives similar results upon simulation. The correlation measures are plotted on the ordinate. The abscissa is the number of computer iterations. A discrete-time network with real-valued parameters was used. The total number of errors in the 14 8-bit binary $\{1, -1\}$ vectors used to initialize the system was 21. This results in an average of 1.5 errors per 8-bit vector. We note that the learning was completed in two time steps. Therefore, in this case at least, we see that the storage requirement is not intensive - only two weight matrices need to be stored during the synthesis phase.

We note that the learning phase by default also represents the autoassociative system error-correcting performance over input samples *drawn from the training set*. Therefore over the training set we can compare this performance with that of the outer product rule (11). By considering corrupted input vectors from the training set, we compare the error-correcting capabilities of the two methods, *not* their capacities to store uncorrupted vectors. In fact we see that the two weight rules become identical when we initialize with the true vectors (this equivalence is not a peculiarity of the new technique, but merely a consequence of the particular performance index *chosen*). In other words, this comparison is a test of the extent of the basins of attraction around the desired memories for the two techniques. Looking at the curves labeled *"Conventional Outer Product"*, we see that the new technique performs better than the outer product rule.

## 4    CONCLUSIONS AND FURTHER WORK

We have described a technique for training neural networks based on formal tools from optimal control theory. For a specific example consisting of learning the input-output map in a training set we derived the relevant weight equations and illustrated the learning phase of the method. This example gives a weight rule that turns out to be a continuous-time generalization of the outer-product rule. Using corrupted vectors from the training set, we show that the new rule performs better in error-correction than the outer-product rule. Simulations on the generalization capabilities of the method are ongoing and are not included in the present work.

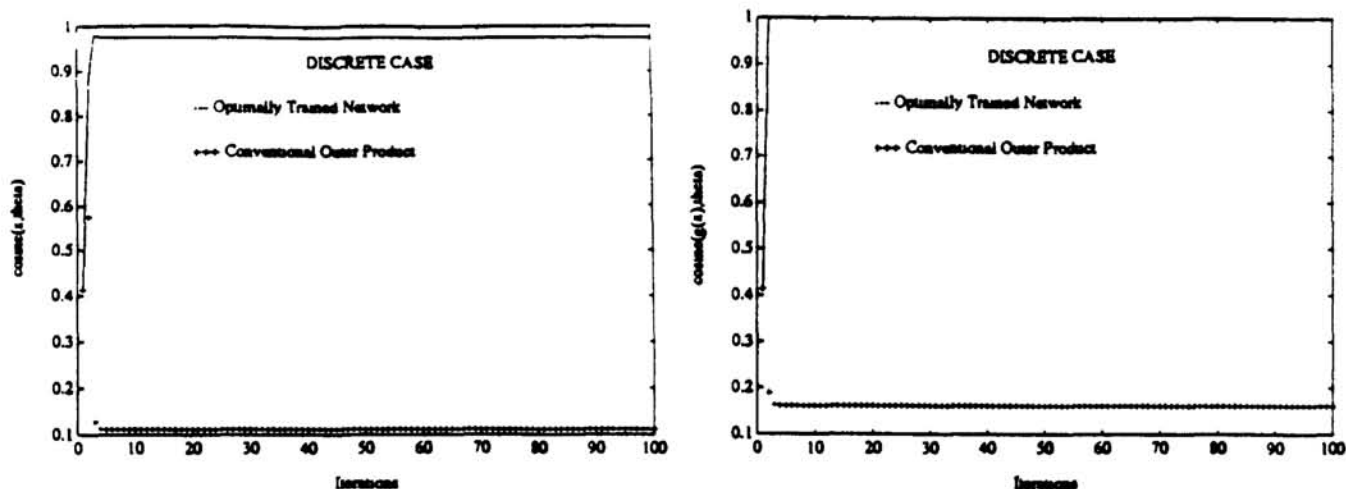

**Figure 1:** Learning Curves

Although we considered a *training set* consisting of input-output vector pairs as the starting point for the procedure, a closer examination shows that this is not required. More generally, what is required is a performance index that reflects the objective of the training. Also in our ongoing work we have extended the results to more general forms of the state equation and the performance index. Using an appropriate performance index we are investigating a network for the Travelling Salesman Problem and related applications like Tracking and Data Association.

## References

[1] Michael A. Cohen & Stephen Grossberg, "Absolute Stability of Global Pattern Formation and Parallel Memory Storage by Competitive Neural Networks," *IEEE Transactions on Systems, Man and Cybernetics* SMC-13 (1983), 815–826.

[2] J. J. Hopfield & D. W. Tank, "Neural Computation of Decisions in Optimization Problems," *Biological Cybernetics* 52 (1985), 141–152.

[3] Arthur E. Bryson & Yu-Chi Ho, *Applied Optimal Control*, Hemisphere, 1975.

[4] J. J. Hopfield, "Neural Networks and Physical Systems with Emergent Collective Computational Abilities," *Proceedings of the National Academy of Sciences* 79 (1982), 2554–2558.
